# A new metric on the manifold of kernel matrices with application to matrix geometric means

**Suvrit Sra**
Max Planck Institute for Intelligent Systems
72076 Tübigen, Germany
`suvrit@tuebingen.mpg.de`

## Abstract

Symmetric positive definite (spd) matrices pervade numerous scientific disciplines, including machine learning and optimization. We consider the key task of measuring distances between two spd matrices; a task that is often nontrivial whenever the distance function must respect the non-Euclidean geometry of spd matrices. Typical non-Euclidean distance measures such as the Riemannian metric $\delta_R(X,Y) = \|\log(Y^{-1/2}XY^{-1/2})\|_\mathrm{F}$, are computationally demanding and also complicated to use. To allay some of these difficulties, we introduce a new metric on spd matrices, which not only respects non-Euclidean geometry but also offers faster computation than $\delta_R$ while being less complicated to use. We support our claims theoretically by listing a set of theorems that relate our metric to $\delta_R(X,Y)$, and experimentally by studying the nonconvex problem of computing matrix geometric means based on squared distances.

## 1 Introduction

Symmetric positive definite (spd) matrices[1] are remarkably pervasive in a multitude of areas, especially machine learning and optimization. Several applications in these areas require an answer to the fundamental question: *how to measure a distance between two spd matrices?*

This question arises, for instance, when optimizing over the set of spd matrices. To judge convergence of an optimization procedure or in the design of algorithms we may need to compute distances between spd matrices [1–3]. As a more concrete example, suppose we wish to retrieve from a large database of spd matrices the "closest" spd matrix to an input query. The quality of such a retrieval depends crucially on the distance function used to measure closeness; a choice that also dramatically impacts the actual search algorithm itself [4, 5]. Another familiar setting is that of computing statistical metrics for multivariate Gaussian distributions [6], or more recently, quantum statistics [7]. Several other applications depend on being able to effectively measure distances between spd matrices–see e.g., [8–10] and references therein.

In many of these domains, viewing spd matrices as members of a Euclidean vector space is *insufficient*, and the non-Euclidean geometry conferred by a suitable metric is of great importance. Indeed, the set of (strict) spd matrices forms a differentiable Riemannian manifold [11, 10] that is perhaps the most studied example of a manifold of nonpositive curvature [12; Ch.10]. These matrices also form a convex cone, and the set of spd matrices in fact serves as a canonical higher-rank symmetric space [13]. The conic view is of great importance in convex optimization [14–16], symmetric spaces are important in algebra and analysis [13, 17], and in optimization [14, 18], while the manifold and other views are also widely important—see e.g., [11; Ch.6] for an overview.

The starting point for this paper is the manifold view. For space reasons, we limit our discussion to $\mathbb{P}(n)$ as a Riemannian manifold, noting that most of the discussion could also be set in terms of Finsler manifolds. But before we go further, let us fix basic notation.

**Notation.** Let $\mathbb{S}_n$ denote the set of $n \times n$ real symmetric matrices. A matrix $A \in \mathbb{S}_n$ is called *positive* (we drop the word "definite" for brevity) if

$$\langle x, Ax \rangle > 0 \quad \text{for all} \quad x \neq 0; \quad \text{also denoted as} \quad A > 0. \tag{1}$$

We denote the set of $n \times n$ positive matrices by $\mathbb{P}_n$. If only the non-strict inequality $\langle x, Ax \rangle \geq 0$ holds (for all $x \in \mathbb{R}^n$) we say $A$ is *positive semidefinite*; this is also denoted as $A \geq 0$. For two matrices $A, B \in \mathbb{S}_n$, the operator inequality $A \geq B$ means that the difference $A - B \geq 0$. The *Frobenius norm* of a matrix $X \in \mathbb{R}^{m \times n}$ is defined as $\|X\|_{\mathrm{F}} = \sqrt{\operatorname{tr}(X^T X)}$, while $\|X\|$ denotes the standard operator norm. For an analytic function $f$ on $\mathbb{C}$, and a diagonalizable matrix $A = U\Lambda U^T$, $f(A) := Uf(\Lambda)U^T$. Let $\lambda(X)$ denote the vector of eigenvalues of $X$ (in any order) and $\operatorname{Eig}(X)$ a diagonal matrix that has $\lambda(X)$ as its diagonal. We also use $\lambda^{\downarrow}(X)$ to denote a sorted (in descending order) version of $\lambda(X)$ and $\lambda^{\uparrow}(X)$ is defined likewise. Finally, we define $\operatorname{Eig}^{\downarrow}(X)$ and $\operatorname{Eig}^{\uparrow}(X)$ as the corresponding diagonal matrices.

**Background.** The set $\mathbb{P}_n$ is a canonical higher-rank symmetric space that is actually an open set within $\mathbb{S}_n$, and thereby a differentiable manifold of dimension $n(n+1)/2$. The tangent space at a point $A \in \mathbb{P}_n$ can be identified with $\mathbb{S}_n$, so a suitable inner-product on $\mathbb{S}_n$ leads to the Riemannian distance on $\mathbb{P}_n$ [11; Ch.6]. At the point $A$ this metric is induced by the differential form

$$ds^2 = \|A^{-1/2}dAA^{-1/2}\|_{\mathrm{F}}^2 = \operatorname{tr}(A^{-1}dAA^{-1}dA). \tag{2}$$

For $A, B \in \mathbb{P}_n$, it is known that there is a unique geodesic joining them given by [11; Thm.6.1.6]:

$$\gamma(t) := A \sharp_t B := A^{1/2}(A^{-1/2}BA^{-1/2})^t A^{1/2}, \quad 0 \leq t \leq 1, \tag{3}$$

and its midpoint $\gamma(1/2)$ is the *geometric mean* of $A$ and $B$. The associated *Riemannian metric* is

$$\delta_R(A, B) := \|\log(A^{-1/2}BA^{-1/2})\|_{\mathrm{F}}, \quad \text{for} \quad A, B > 0. \tag{4}$$

From definition (4) it is apparent that computing $\delta_R$ will be computationally demanding, and requires care. Indeed, to compute (4) we must essentially compute generalized eigenvalues of $A$ and $B$. For an application that must repeatedly compute distances between numerous pairs of matrices this computational burden can be excessive [4]. Driven by such computational concerns, Cherian et al. [4] introduced a symmetrized "log-det" based *matrix divergence*:

$$J(A, B) = \log\det\left(\tfrac{A+B}{2}\right) - \tfrac{1}{2}\log\det(AB) \quad \text{for} \quad A, B > 0. \tag{5}$$

This divergence was used as a proxy for $\delta_R$ and observed that $J(A, B)$ offers the same level of performance on a difficult nearest neighbor retrieval task as $\delta_R$, while being many times faster! Among other reasons, a large part of their speedup was attributed to the avoidance of eigenvalue computations for obtaining $J(A, B)$ or its derivatives, a luxury the $\delta_R$ does not permit. Independently, Chebbi and Moahker [2] also introduced a slightly generalized version of (5) and studied some of its properties, especially computation of "centroids" of positive matrices using their matrix divergence.

Interestingly, Cherian et al. [4] claimed that $\sqrt{J(A, B)}$ might *not* be metric, whereas Chebbi and Moahker [2] *conjectured* that $\sqrt{J(A, B)}$ *is* a metric. We resolve this uncertainty and prove that $\sqrt{J(A, B)}$ is indeed a metric, albeit not one that embeds isometrically into a Hilbert space.

Due to space constraints, we only summarily mention several of the properties that this metric satisfies, primarily to help develop intuition that motivates $\sqrt{J}$ as a good proxy for the Riemannian metric $\delta_R$. We apply these insights to study *"matrix geometric means"* of set of positive matrices: a problem also studied in [4, 2]. Both cited papers have some gaps in their claims, which we fill by proving that even though computing the geometric mean is a nonconvex problem, we can still compute it efficiently and optimally.

## 2  The $\delta_{\ell d}$ metric

The main result of this paper is Theorem 1.

**Theorem 1.** *Let $J$ be as in (5), and define $\delta_{\ell d} := \sqrt{J}$. Then, $\delta_{\ell d}$ is a metric on $\mathbb{P}_n$.*

Our proof of Theorem 1 depends on several key steps. Due to restrictions on space we cannot include full proofs of all the results, and refer the reader to the longer article [19] instead. We do, however, provide sketches for the crucial steps in our proof.

**Proposition 2.** *Let $A, B, C \in \mathbb{P}_n$. Then, (i) $\delta_{\ell d}(I, A) = \delta_{\ell d}(I, \mathrm{Eig}(A))$; (ii) for $P, Q \in GL(n, \mathbb{C})$, $\delta_{\ell d}(PAQ, PBQ) = \delta_{\ell d}(A, B)$; (iii) for $X \in GL(n, \mathbb{C})$, $\delta_{\ell d}(X^*AX, X^*BX) = \delta_{\ell d}(A, B)$; (iv) $\delta_{\ell d}(A, B) = \delta_{\ell d}(A^{-1}, B^{-1})$; (v) $\delta_{\ell d}(A \otimes B, A \otimes C) = \sqrt{n}\delta_{\ell d}(B, C)$, where $\otimes$ denotes the Kronecker or tensor product.*

The first crucial result is that for positive scalars, $\delta_{\ell d}$ is indeed a metric. To prove this, recall the notion of *negative definite* functions (Def. 3), and a related classical result of Schoenberg (Thm. 4).

**Definition 3** ([20; Def. 1.1]). *Let $\mathcal{X}$ be a nonempty set. A function $\psi : \mathcal{X} \times \mathcal{X} \to \mathbb{R}$ is said to be* negative definite *if for all $x, y \in \mathcal{X}$ it is symmetric ($\psi(x, y) = \psi(y, x)$), and satisfies the inequality*

$$\sum_{i,j=1}^{n} c_i c_j \psi(x_i, x_j) \leq 0, \tag{6}$$

*for all integers $n \geq 2$, and subsets $\{x_i\}_{i=1}^{n} \subseteq \mathcal{X}$, $\{c_i\}_{i=1}^{n} \subseteq \mathbb{R}$ with $\sum_{i=1}^{n} c_i = 0$.*

**Theorem 4** [20; Prop. 3.2, Chap. 3]). *Let $\psi : \mathcal{X} \times \mathcal{X} \to \mathbb{R}$ be negative definite. Then, there is a Hilbert space $\mathcal{H} \subseteq \mathbb{R}^{\mathcal{X}}$ and a mapping $x \mapsto \varphi(x)$ from $\mathcal{X} \to \mathcal{H}$ such that we have the equality*

$$\|\varphi(x) - \varphi(y)\|_{\mathcal{H}}^2 = \psi(x, y) - \tfrac{1}{2}(\psi(x, x) + \psi(y, y)). \tag{7}$$

*Moreover, negative definiteness of $\psi$ is necessary for such a mapping to exist.*

**Theorem 5** (Scalar case). *Define $\delta_s^2(x, y) := \log[(x + y)/(2\sqrt{xy})]$ for scalars $x, y > 0$. Then,*

$$\delta_s(x, y) \leq \delta_s(x, z) + \delta_s(y, z) \quad \text{for all } x, y, z > 0. \tag{8}$$

*Proof.* We show that $\psi(x, y) = \log\left(\frac{x+y}{2}\right)$ is negative definite. Since $\delta_s^2(x, y) = \psi(x, y) - \frac{1}{2}(\psi(x, x) + \psi(y, y))$, Thm. 4 then implies the triangle inequality (8). To prove $\psi$ is negative definite, by [Thm. 2.2, Chap. 3, 20] we may equivalently show that $e^{-\beta\psi(x,y)} = ((x + y)/2)^{-\beta}$ is a positive definite function for $\beta > 0$, and all $x, y > 0$. To that end, it suffices to show that the matrix

$$H = [h_{ij}] = \left[(x_i + x_j)^{-\beta}\right], \quad 1 \leq i, j \leq n,$$

is positive definite for every integer $n \geq 1$, and positive numbers $\{x_i\}_{i=1}^{n}$. Now, observe that

$$h_{ij} = \frac{1}{(x_i + x_j)^\beta} = \frac{1}{\Gamma(\beta)} \int_0^\infty e^{-t(x_i + x_j)} t^{\beta-1} dt, \tag{9}$$

where $\Gamma(\beta) = \int_0^\infty e^{-t} t^{\beta-1} dt$ is the well-known Gamma function. Thus, with $f_i(t) = e^{-tx_i} t^{\frac{\beta-1}{2}} \in L_2([0, \infty))$, we see that $[h_{ij}]$ equals the Gram matrix $[\langle f_i, f_j \rangle]$, whereby $H > 0$. $\square$

Using Thm. 5 we obtain the following simple but important "Minkowsi" inequality for $\delta_s$.

**Corollary 6.** *Let $x, y, z > 0$ be scalars, and let $p \geq 1$. Then,*

$$\left(\sum_{i=1}^{n} \delta_s^p(x_i, y_i)\right)^{1/p} \leq \left(\sum_{i=1}^{n} \delta_s^p(x_i, z_i)\right)^{1/p} + \left(\sum_{i=1}^{n} \delta_s^p(y_i, z_i)\right)^{1/p}. \tag{10}$$

**Corollary 7.** *Let $X, Y, Z > 0$ be diagonal matrices. Then,*

$$\delta_{\ell d}(X, Y) \leq \delta_{\ell d}(X, Z) + \delta_{\ell d}(Y, Z) \tag{11}$$

Next, we recall a fundamental determinantal inequality.

**Theorem 8** ([21; Exercise VI.7.2]). *Let $A, B \in \mathbb{P}_n$. Then,*

$$\prod_{i=1}^{n} (\lambda_i^\downarrow(A) + \lambda_i^\downarrow(B)) \leq \det(A + B) \leq \prod_{i=1}^{n} (\lambda_i^\downarrow(A) + \lambda_i^\uparrow(B)). \tag{12}$$

**Corollary 9.** *Let $A, B > 0$. Then,*

$$\delta_{\ell d}(\text{Eig}^{\downarrow}(A), \text{Eig}^{\downarrow}(B)) \quad \leq \quad \delta_{\ell d}(A, B) \quad \leq \quad \delta_{\ell d}(\text{Eig}^{\downarrow}(A), \text{Eig}^{\uparrow}(B))$$

The final result that we need is a well-known fact from linear algebra (our own proof is in [19]).

**Lemma 10** ([e.g., 22; p.58]). *Let $A > 0$, and let $B$ be Hermitian. There is a matrix $P$ for which*

$$P^*AP = I, \quad and \quad P^*BP = D, \quad and\ D\ is\ diagonal. \tag{13}$$

With all these theorems and lemmas in hand, we are now finally ready to prove Thm. 1.

*Proof.* (Theorem 1). We must prove that $\delta_{\ell d}$ is symmetric, nonnegative, definite, and that is satisfies the triangle inequality. Symmetry is immediate from definition. Nonnegativity and definiteness follow from the strict log-concavity (on $\mathbb{P}_n$) of the determinant, whereby

$$\det\left(\frac{X+Y}{2}\right) \geq \det(X)^{1/2} \det(Y)^{1/2},$$

which equality iff $X = Y$, which in turn implies that $\delta_{\ell d}(X, Y) \geq 0$ with equality iff $X = Y$. The only hard part is to prove the triangle inequality, a result that has eluded previous attempts [4, 2].

Let $X, Y, Z > 0$ be arbitrary. From Lemma 10 we know that there is a matrix $P$ such that $P^*XP = I$ and $P^*YP = D$. Since $Z > 0$ is arbitrary, and congruence preserves positive definiteness, we may write just $Z$ instead of $P^*ZP$. Also, since $\delta_{\ell d}(P^*XP, P^*YP) = \delta_{\ell d}(X, Y)$ (see Prop. 2), proving the triangle inequality reduces to showing that

$$\delta_{\ell d}(I, D) \leq \delta_{\ell d}(I, Z) + \delta_{\ell d}(D, Z). \tag{14}$$

Consider now the diagonal matrices $D^{\downarrow}$ and $\text{Eig}^{\downarrow}(Z)$. Corollary 7 asserts the inequality

$$\delta_{\ell d}(I, D^{\downarrow}) \leq \delta_{\ell d}(I, \text{Eig}^{\downarrow}(Z)) + \delta_{\ell d}(D^{\downarrow}, \text{Eig}^{\downarrow}(Z)). \tag{15}$$

Prop. 2(i) implies that $\delta_{\ell d}(I, D) = \delta_{\ell d}(I, D^{\downarrow})$ and $\delta_{\ell d}(I, Z) = \delta_{\ell d}(I, \text{Eig}^{\downarrow}(Z))$, while Cor. 9 shows that $\delta_{\ell d}(D^{\downarrow}, \text{Eig}^{\downarrow}(Z)) \leq \delta_{\ell d}(D, Z)$. Combining these inequalities, we obtain (14), as desired. $\square$

Although the metric space $(\mathbb{P}_n, \delta_{\ell d})$ has numerous fascinating properties, due to space concerns, we do not discuss it further. Instead we discuss a connection more important to machine learning and related areas: kernel functions arising from $\delta_{\ell d}$. Indeed, some of connections (e.g., Thm. 11) have already been successfully applied very recently in computer vision [23].

## 2.1 Hilbert space embedding of $\delta_{\ell d}$

Theorem 1 shows that $\delta_{\ell d}$ is a metric and Theorem 5 shows that actually for positive scalars, the metric space $(\mathbb{R}_{++}, \delta_s)$ embeds isometrically into a Hilbert space. It is, therefore, natural to ask whether $(\mathbb{P}_n, \delta_{\ell d})$ also admits such an embedding?

Theorem 4 says that such a kernel exists if and only if $\delta_{\ell d}^2$ is negative definite; equivalently, iff

$$e^{-\beta \delta_{\ell d}^2(X,Y)} = \frac{\det(XY)^{\beta}}{\det((X+Y)/2)^{\beta}}, \tag{16}$$

is a positive definite kernel for all $\beta > 0$. To verify this, it suffices to check if the matrix

$$H_{\beta} = [h_{ij}] := \left[\frac{1}{\det(X_i + X_j)^{\beta}}\right], \quad 1 \leq i, j \leq m, \tag{17}$$

is positive for every integer $m \geq 1$ and arbitrary positive matrices $X_1, \ldots, X_m$.

Unfortunately, a numerical experiment (see [19]) reveals that $H_{\beta}$ is not always positive. This implies that $(\mathbb{P}_d, \delta_{\ell d})$ cannot embed isometrically into a Hilbert space. Undeterred, we still ask: *For what choices of $\beta$ is $H_{\beta}$ positive?* Surprisingly, this question admits a complete answer. Theorem 11 characterizes the values of $\beta$ necessary and sufficient for $H_{\beta}$ to be positive. We note here that the case $\beta = 1$ was essentially treated in [24], in the context of semigroup kernels on measures.

**Theorem 11.** *Let $X_1, \ldots, X_m \in \mathbb{P}_n$. The matrix $H_{\beta}$ defined by (17) is positive, if and only if*

$$\beta \in \left\{\tfrac{j}{2} : j \in \mathbb{N}, \ and\ 1 \leq j \leq (n-1)\right\} \cup \left\{\gamma : \gamma \in \mathbb{R}, and\ \gamma > \tfrac{1}{2}(n-1)\right\}. \tag{18}$$

*Proof.* We first prove the "if" part. Define the function $f_i := \frac{1}{\pi^{n/4}} e^{-x^T X_i x}$ (for $1 \leq i \leq m$). Then, $f_i \in L_2(\mathbb{R}^n)$, where the inner-product is given by the Gaussian integral

$$\langle f_i, f_j \rangle := \frac{1}{\pi^{d/2}} \int_{\mathbb{R}^n} e^{-x^T(X_i + X_j)x} dx = \frac{1}{\det(X_i + X_j)^{1/2}}. \tag{19}$$

From (19) it follows that $H_{1/2}$ is positive. Since the Schur (elementwise) product of two positive matrices is again positive, it follows that $H_\beta > 0$ whenever $\beta$ is an integer multiple of $1/2$. To extend the result to all $\beta$ covered by (18), we need a more intricate integral representation, namely the *multivariate Gamma function*, defined as [25; §2.1.2]

$$\Gamma_n(\beta) := \int_{\mathbb{P}_n} e^{-\operatorname{tr}(A)} \det(A)^{\beta - (n+1)/2} dA, \tag{20}$$

where the integral converges for $\beta > \frac{1}{2}(n-1)$. Define for each $i$ the function $f_i := ce^{-\operatorname{tr}(AX_i)}$ ($c > 0$ is a constant). Then, $f_i \in L_2(\mathbb{P}_n)$, which we equip with the inner product

$$\langle f_i, f_j \rangle := c^2 \int_{\mathbb{P}_n} e^{-\operatorname{tr}(A(X_i + X_j))} \det(A)^{\beta - (n+1)/2} dA = \det(X_i + X_j)^{-\beta},$$

and it exists whenever $\beta > \frac{1}{2}(n-1)$. Consequently, $H_\beta$ is positive for all $\beta$ defined by (18).

The "only if" part follows from deeper results in the rich theory of symmetric spaces.[2] Specifically, since $\mathbb{P}_n$ is a symmetric cone, and $1/\det(X)$ is a decreasing function on this cone, (i.e., $1/\det(X + Y) \leq 1/\det(X)$ for all $X, Y > 0$), an appeal to [26; VII.3.1] grants our claim. □

**Remark 12.** *Readers versed in stochastic processes will recognize that the above result provides a different perspective on a classical result concerning* infinite divisibility *of Wishart processes [27], where the set* (18) *also arises as a consequence of Gindikin's theorem [28].*

At this point, it is worth mentioning the following "obvious" result.

**Theorem 13.** *Let $\mathcal{X}$ be a set of positive matrices that commute with each other. Then, $(\mathcal{X}, \delta_{\ell d})$ can be isometrically embedded into some Hilbert space.*

*Proof.* The proof follows because a commuting set of matrices can be simultaneously diagonalized, and for diagonal matrices, $\delta_{\ell d}^2(X, Y) = \sum_i \delta_s^2(X_{ii}, Y_{ii})$, which is a nonnegative sum of negative definite kernels and is therefore itself negative definite. □

## 3 Connections between $\delta_{\ell d}$ and $\delta_R$

After showing that $\delta_{\ell d}$ is a metric and studying its relation to kernel functions, let us now return to our original motivation: introducing $\delta_{\ell d}$ as a reasonable alternative to the widely used Riemannian metric $\delta_R$. We note here that Cherian et al. [4; 29] offer strong experimental evidence supporting $\delta_{\ell d}$ as an alternative; we offer more theoretical results.

Our theoretical results are based around showing that $\delta_{\ell d}$ fulfills several properties akin to those displayed by $\delta_R$. Due to lack of space, we present only a summary of our results in Table 1, and cite the corresponding theorems in the longer article [19] for proofs. While the actual proofs are valuable and instructive, the key message worth noting is: *both $\delta_R$ and $\delta_{\ell d}$ express the (negatively curved) non-Euclidean geometry of their respective metric spaces by displaying similar properties.*

## 4 Application: computing geometric means

In this section we turn our attention to an object that perhaps connects $\delta_R$ and $\delta_{\ell d}$ most intimately: the operator *geometric mean* (GM), which is given by the midpoint of the geodesic (3), denoted as

$$A\sharp B := \gamma(1/2) = A^{1/2}(A^{-1/2} B A^{-1/2})^{1/2} A^{1/2}. \tag{21}$$

| Riemannian metric | Ref. | $\delta_{\ell d}$-metric | Ref. |
|---|---|---|---|
| $\delta_R(X^*AX, X^*BX) = \delta_R(A,B)$ | [11; Ch.6] | $\delta_{\ell d}(X^*AX, X^*BX) = \delta_{\ell d}(A,B)$ | Prop. 2 |
| $\delta_R(A^{-1}, B^{-1}) = \delta_R(A,B)$ | [11; Ch.6] | $\delta_{\ell d}(A^{-1}, B^{-1}) = \delta_{\ell d}(A,B)$ | Prop. 2 |
| $\delta_R(A^t, B^t) \leq t\delta_R(A,B)$ | [11; Ex.6.5.4] | $\delta_{\ell d}(A^t, B^t) \leq \sqrt{t}\delta_{\ell d}(A,B)$ | [19; Th.4.6] |
| $\delta_R(A^s, B^s) \leq (s/u)\delta_R(A^u, B^u)$ | [19; Th.4.11] | $\delta_{\ell d}(A^s, B^s) \leq \sqrt{s/u}\delta_{\ell d}(A^u, B^u)$ | [19; Th.4.11] |
| $\delta_R(A, A\sharp B) = \delta_R(B, A\sharp B)$ | Trivial | $\delta_{\ell d}(A, A\sharp B) = \delta_{\ell d}(B, A\sharp B)$ | Th.14 |
| $\delta_R(A, A\sharp_t B) = t\delta_R(A,B)$ | [11; Th.6.1.6] | $\delta_{\ell d}(A, A\sharp_t B) \leq \sqrt{t}\delta_{\ell d}(A,B)$ | [19; Th.4.7] |
| $\delta_R(A\sharp_t B, A\sharp_t C) \leq t\delta_R(B,C)$ | [11; Th.6.1.2] | $\delta_{\ell d}(A\sharp_t B, A\sharp_t C) \leq \sqrt{t}\delta_{\ell d}(B,C)$ | [19; Th.4.8] |
| $\delta_R^2(X,A) + \delta_R^2(X,B) \overset{\min}{\mapsto} \text{GM}$ | [11; Ch. 6] | $\delta_{\ell d}^2(X,A) + \delta_{\ell d}^2(X,B) \overset{\min}{\mapsto} \text{GM}$ | Th.14 |
| $\delta_R(A+X, A+Y) \leq \delta_R(X,Y)$ | [3] | $\delta_{\ell d}(A+X, A+Y) \leq \delta_{\ell d}(X,Y)$ | [19; Th.4.9] |

Table 1: Some of the similarities between $\delta_R$ and $\delta_{\ell d}$. All matrices are assumed to be in $\mathbb{P}_n$. The scalars $t, s, u$ satisfy $0 < t \leq 1, 1 \leq s \leq u < \infty$.

The GM (21) has numerous attractive properties—see for instance [30]—among these, the following variational characterization is very important [31, 32],

$$A\sharp B = \text{argmin}_{X>0} \quad \delta_R^2(A,X) + \delta_R^2(B,X). \tag{22}$$

especially because it generalizes the matrix geometric mean to more than two matrices. Specifically, this "natural" generalization is the *Karcher mean* (Fréchet mean) [31, 32, 11]:

$$GM(A_1, \ldots, A_m) := \text{argmin}_{X>0} \quad \sum_{i=1}^m \delta_R^2(X, A_i). \tag{23}$$

This multivariable generalization is in fact a well-studied difficult problem—see e.g., [33] for information on state-of-the-art. Indeed, its inordinate computational expenses motivated Cherian et al. [4] to study the alternative mean

$$GM_{\ell d}(A_1, \ldots, A_m) := \underset{X>0}{\text{argmin}} \quad \phi(X) := \sum_{i=1}^m \delta_{\ell d}^2(X, A_i), \tag{24}$$

which has also been more thoroughly studied by Chebbi and Moahker [2].

Although the mean (24) was previously studied in [4, 2], some crucial aspects were missing. Specifically, Cherian et al. [4] only proved their solution to be a stationary point of $\phi(X)$; they did *not prove* either global or local optimality. Although Chebbi and Moahker [2] showed that (24) has a unique solution, like [4] they too *only proved stationarity*, neither global nor local optimality.

We fill these gaps, and we make the following main contributions below:

1. We connect (24) to the Karcher mean more closely, where in Theorem 14 we shows that for the two matrix case both problems have the same solution;
2. We show that the unique positive solution to (24) is globally optimal; this result is particularly interesting because $\phi(X)$ is nonconvex.

We begin by looking at the two variable case of $GM_{\ell d}$ (24).

**Theorem 14.** *Let* $A, B > 0$. *Then,*

$$A\sharp B = \text{argmin}_{X>0} \quad \phi(X) := \delta_{\ell d}^2(X, A) + \delta_{\ell d}^2(X, B). \tag{25}$$

*Moreover,* $A\sharp B$ *is equidistant from* $A$ *and* $B$, *i.e.,* $\delta_{\ell d}(A, A\sharp B) = \delta_{\ell d}(B, A\sharp B)$.

*Proof.* If $A = B$, then clearly $X = A$ minimizes $\phi(X)$. Assume therefore, that $A \neq B$. Ignoring the constraint $X > 0$ momentarily, we see that any stationary point must satisfy $\nabla\phi(X) = 0$. Thus,

$$\nabla\phi(X) = \left(\frac{X+A}{2}\right)^{-1}\frac{1}{2} + \left(\frac{X+B}{2}\right)^{-1}\frac{1}{2} - X^{-1} = 0$$
$$\implies (X+A)X^{-1}(X+B) = 2X + A + B \implies B = XA^{-1}X. \tag{26}$$

The latter equation is a Riccati equation that is known to have a *unique, positive definite* solution given by the matrix GM (21) (see [11; Prop 1.2.13]). All that remains to show is that this GM is in fact a local minimizer. To that end, we must show that the Hessian $\nabla^2\phi(X) > 0$ at $X = A\sharp B$; but this claim is immediate from Theorem 18. So $A\sharp B$ is a *strict* local minimum of (8), which is actually a global minimum because it is the unique positive solution to $\phi(X) = 0$. Finally, the equidistance property follows after some algebraic manipulations; we omit details for brevity [19]. □

Let us now turn to the general case (24). The first-order optimality condition is

$$\nabla\phi(X) = \sum_{i=1}^{m} \tfrac{1}{2} \left(\tfrac{X+A_i}{2}\right)^{-1} - \tfrac{1}{2}mX^{-1} = 0, \qquad X > 0. \tag{27}$$

From (27) using Lemma 15 it can be inferred that [see also 2, 4] that any critical point $X$ of (24) lies in a convex, compact set specified by $\left(\tfrac{1}{m}\sum_{i=1}^{m} A_i^{-1}\right)^{-1} \preceq X \preceq \left(\tfrac{1}{m}\sum_{i=1}^{m} A_i\right)$.

**Lemma 15** ([21; Ch.5]). *The map $X^{-1}$ on $\mathbb{P}_n$ is order reversing and operator convex. That is, for $X, Y \in \mathbb{P}_n$, if $X \geq Y$, then $X^{-1} \leq Y^{-1}$; for $t \in [0,1]$, $(tX + (1-t)Y)^{-1} \leq tX^{-1} + (1-t)Y^{-1}$.*

**Lemma 16** ([19]). *Let $A, B, C, D \in \mathbb{P}_n$, so that $A \geq B$ and $C \geq D$. Then, $A \otimes C \geq B \otimes D$.*

**Lemma 17** (Uniqueness [2]). *The nonlinear equation (27) has a unique positive solution.*

Using the above results, we can finally prove the main theorem of this section.

**Theorem 18.** *Let $X$ be a matrix satisfying* (27). *Then, it is the unique global minimizer of* (24).

*Proof.* The objective function $\phi(X)$ (24) has only one positive stationary point, which follows from Lemma 17. Let $X$ be this stationary point satisfying (27). We show that $X$ is actually a local minimum; global optimality is immediate from uniqueness of $X$.

To show local optimality, we prove that the Hessian $\nabla^2\phi(X) > 0$. Ignoring constants, showing positivity of the Hessian reduces to proving that

$$mX^{-1} \otimes X^{-1} - \sum_{i=1}^{m} \tfrac{1}{2} \left(\tfrac{X+A_i}{2}\right)^{-1} \otimes \left(\tfrac{X+A_i}{2}\right)^{-1} > 0. \tag{28}$$

Now replace $mX^{-1}$ in (28) using the condition (27); therewith inequality (28) turns into

$$\left(\sum_{i=1}^{m} \left(\tfrac{X+A_i}{2}\right)^{-1}\right) \otimes X^{-1} > \sum_{i=1}^{m} \left(\tfrac{X+A_i}{2}\right)^{-1} \otimes (X + A_i)^{-1}$$
$$\Longleftrightarrow \quad \sum_{i=1}^{m} \left(\tfrac{X+A_i}{2}\right)^{-1} \otimes X^{-1} > \sum_{i=1}^{m} \left(\tfrac{X+A_i}{2}\right)^{-1} \otimes (X + A_i)^{-1}. \tag{29}$$

From Lemma 15 we know that $X^{-1} > (X + A_i)^{-1}$, so that an application of Lemma 16 shows that $\left(\tfrac{X+A_i}{2}\right)^{-1} \otimes X^{-1} > \left(\tfrac{X+A_i}{2}\right)^{-1} \otimes (X + A_i)^{-1}$ for $1 \leq i \leq m$. Summing up, we obtain (29), which implies the desired local (and by uniqueness, global) optimality of $X$. $\qquad\square$

**Remark 19.** It is worth noting that Theorem 18 establishes that solving (27) yields the *global minimum* of a nonconvex optimization problem. This result is even more remarkable because unlike CAT(0)-metrics such as $\delta_R$, the metric $\delta_{\ell d}$ is *not* geodesically convex.

### 4.1 Numerical Results

We present a key numerical result to illustrate the large savings in running time when computing with $\delta_{\ell d}$ when compared with $\delta_R$. To compute the Karcher mean we downloaded the "Matrix Means Toolbox" of Bini and Iannazzo from *http://bezout.dm.unipi.it/software/mmtoolbox/*. In particular, we use the file called `rich.m` which implements a state-of-the-art method [33].

The first plot in Fig. 1 indicate that $\delta_{\ell d}$ can be around 5 times faster than $\delta_{R2}$ and up to 50 times faster than $\delta_{R1}$. The second plot shows how expensive it can be to compute $GM$ (23) as opposed to $GM_{\ell d}$ (24)—up to 1000 times! The former was computed using the method of [33], while the latter runs the fixed-point iteration proposed in [2] (the iteration was run until $\|\nabla\phi(X)\|$ fell below $10^{-10}$). The key point here is not that the fixed-point iteration is faster, but rather that (24) is a much simpler problem thanks to the convenient eigenvalue free structure of $\delta_{\ell d}$.

## 5  Conclusions and future work

We presented a new metric on the manifold of positive definite matrices, and related it to the classical Riemannian metric on this manifold. Empirically, our new metric was shown to lead to large computational gains, while theoretically, a series of theorems demonstrated how it expresses the negatively curved non-Euclidean geometry in a manner analogous to the Riemannian metric.

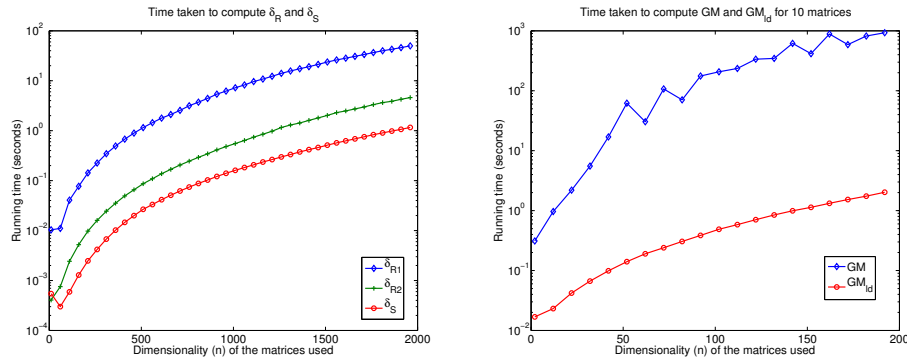

Figure 1: Running time comparisons between $\delta_R$ and $\delta_{\ell d}$. The left panel shows time (in seconds) taken to compute $\delta_R$ and $\delta_{\ell d}$, averaged over 10 runs to reduce variance. In the plot, $\delta_{R1}$ refers to the implementation of $\delta_R$ in the matrix means toolbox [33], while $\delta_{R2}$ is our own implementation.

At this point, there are several directions of future work opened by our paper. We mention some of the most relevant ones below. (i) Study further geometric properties of the metric space $(\mathbb{P}_n, \delta_{\ell d})$; (ii) Further enrich the connections to $\delta_R$, and to other (Finsler) metrics on $\mathbb{P}_n$; (iii) Study properties of geometric mean $GM_{\ell d}$ (24), including faster algorithms to compute it; (iv) Akin to [4], apply $\delta_{\ell d}$ in where $\delta_R$ has been so far dominant. We plan to tackle some of these problems, and hope that our paper encourages other researchers in machine learning and optimization to also study them.

## Footnotes

[1]We could equally consider Hermitian matrices, but for simplicity we consider only real matrices.

[2]Specifically, the set (18) is identical to the *Wallach set* which is important in the study of Hilbert spaces of holomorphic functions over symmetric domains [26; Ch.XIII].

# References

[1] H. Lee and Y. Lim. Invariant metrics, contractions and nonlinear matrix equations. *Nonlinearity*, 21: 857–878, 2008.

[2] Z. Chebbi and M. Moahker. Means of hermitian positive-definite matrices based on the log-determinant $\alpha$-divergence function. *Linear Algebra and its Applications*, 436:1872–1889, 2012.

[3] P. Bougerol. Kalman Filtering with Random Coefficients and Contractions. *SIAM J. Control Optim.*, 31 (4):942–959, 1993.

[4] A. Cherian, S. Sra, A. Banerjee, and N. Papanikolopoulos. Efficient Similarity Search for Covariance Matrices via the Jensen-Bregman LogDet Divergence. In *International Conference on Computer Vision (ICCV)*, Nov. 2011.

[5] F. Porikli, O. Tuzel, and P. Meer. Covariance Tracking using Model Update Based on Lie Algebra. In *IEEE CVPR*, 2006.

[6] L. T. Skovgaard. A Riemannian Geometry of the Multivariate Normal Model. *Scandinavian Journal of Statistics*, 11(4):211–223, 1984.

[7] D. Petz. *Quantum Information Theory and Quantum Statistics*. Springer, 2008.

[8] I. Dryden, A. Koloydenko, and D. Zhou. Non-Euclidean statistics for covariance matrices, with applications to diffusion tensor imaging. *Annals of Applied Statistics*, 3(3):1102–1123, 2009.

[9] H. Zhu, H. Zhang, J. G. Ibrahim, and B. S. Peterson. Statistical Analysis of Diffusion Tensors in Diffusion-Weighted Magnetic Resonance Imaging Data. *Journal of the American Statistical Association*, 102(480): 1085–1102, 2007.

[10] F. Hiai and D. Petz. Riemannian metrics on positive definite matrices related to means. *Linear Algebra and its Applications*, 430:3105–3130, 2009.

[11] R. Bhatia. *Positive Definite Matrices*. Princeton University Press, 2007.

[12] M. R. Bridson and A. Haeflinger. *Metric Spaces of Non-Positive Curvature*. Springer, 1999.

[13] A. Terras. *Harmonic Analysis on Symmetric Spaces and Applications*, volume II. Springer, 1988.

[14] Yu. Nesterov and A. Nemirovskii. *Interior-Point Polynomial Algorithms in Convex Programming*. SIAM, 1987.

[15] A. Ben-Tal and A. Nemirovksii. *Lectures on modern convex optimization: Analysis, algorithms, and engineering applications*. SIAM, 2001.

[16] Yu. Nesterov and M. J. Todd. On the riemannian geometry defined for self-concordant barriers and interior point methods. *Found. Comput. Math.*, 2:333–361, 2002.

[17] S. Helgason. *Geometric Analysis on Symmetric Spaces*. Number 39 in Mathematical Surveys and Monographs. AMS, second edition, 2008.

[18] H. Wolkowicz, R. Saigal, and L. Vandenberghe, editors. *Handbook of Semidefinite Programming: Theory, Algorithms, and Applications*. Kluwer Academic, 2000.

[19] S. Sra. Positive definite matrices and the Symmetric Stein Divergence. *arXiv: 1110.1773*, October 2012.

[20] C. Berg, J. P. R. Christensen, and P. Ressel. *Harmonic analysis on semigroups: theory of positive definite and related functions*, volume 100 of *GTM*. Springer, 1984.

[21] R. Bhatia. *Matrix Analysis*. Springer, 1997.

[22] R. Bellman. *Introduction to Matrix Analysis*. SIAM, second edition, 1970.

[23] M. Harandi, C. Sanderson, R. Hartley, and B. Lovell. Sparse Coding and Dictionary Learning for Symmetric Positive Definite Matrices: A Kernel Approach. In *European Conference on Computer Vision (ECCV)*, 2012.

[24] M. Cuturi, K. Fukumizu, and J. P. Vert. Semigroup kernels on measures. *JMLR*, 6:1169–1198, 2005.

[25] R. J. Muirhead. *Aspects of multivariate statistical theory*. Wiley Interscience, 1982.

[26] J. Faraut and A. Korányi. *Analysis on Symmetric Cones*. Clarendon Press, 1994.

[27] M.-F. Bru. Wishart Processes. *J. Theoretical Probability*, 4(4), 1991.

[28] S. G. Gindikin. Invariant generalized functions in homogeneous domains. *Functional Analysis and its Applications*, 9:50–52, 1975.

[29] A. Cherian, S. Sra, A. Banerjee, and N. Papanikolopoulos. Jensen-Bregman LogDet Divergence with Application to Efficient Similarity Search for Covariance Matrices. *IEEE TPAMI*, 2012. *Submitted.*

[30] T. Ando. Concavity of certain maps on positive definite matrices and applications to hadamard products. *Linear Algebra and its Applications*, 26(0):203–241, 1979.

[31] R. Bhatia and J. A. R. Holbrook. Riemannian geometry and matrix geometric means. *Linear Algebra Appl.*, 413:594–618, 2006.

[32] M. Moakher. A differential geometric approach to the geometric mean of symmetric positive-definite matrices. *SIAM J. Matrix Anal. Appl. (SIMAX)*, 26:735–747, 2005.

[33] D. A. Bini and B. Iannazzo. Computing the Karcher mean of symmetric positive definite matrices. *Linear Algebra and its Applications*, Oct. 2011. Available online.

